# What Makes Some POMDP Problems Easy to Approximate?

**David Hsu**[*]     **Wee Sun Lee**[*]     **Nan Rong**[†]

[*]*Department of Computer Science*
*National University of Singapore*
*Singapore, 117590, Singapore*

[†]*Department of Computer Science*
*Cornell University*
*Ithaca, NY 14853, USA*

## Abstract

Point-based algorithms have been surprisingly successful in computing approximately optimal solutions for partially observable Markov decision processes (POMDPs) in high dimensional belief spaces. In this work, we seek to understand the belief-space properties that allow some POMDP problems to be approximated efficiently and thus help to explain the point-based algorithms' success often observed in the experiments. We show that an approximately optimal POMDP solution can be computed in time polynomial in the covering number of a reachable belief space, which is the subset of the belief space reachable from a given belief point. We also show that under the weaker condition of having a small covering number for an optimal reachable space, which is the subset of the belief space reachable under an optimal policy, computing an approximately optimal solution is NP-hard. However, given a suitable set of points that "cover" an optimal reachable space well, an approximate solution can be computed in polynomial time. The covering number highlights several interesting properties that reduce the complexity of POMDP planning in practice, *e.g.*, fully observed state variables, beliefs with sparse support, smooth beliefs, and circulant state-transition matrices.

## 1   Introduction

Computing an optimal policy for a partially observable Markov decision process (POMDP) is an intractable problem [10, 9]. Intuitively, the intractability is due to the "curse of dimensionality": the belief space $\mathcal{B}$ used in solving a POMDP typically has dimensionality equal to $|S|$, the number of states in the POMDP, and therefore the size of $\mathcal{B}$ grows exponentially with $|S|$. As a result, the number of states is often used in practice as an important measure of the complexity of POMDP planning. However, in recent years, point-based POMDP algorithms have made impressive progress in computing approximate solutions by *sampling* the belief space: POMDPs with hundreds of states have been solved in a matter of seconds [14, 4]. It seems surprising that even an approximate solution can be obtained in seconds in a space of hundreds of dimensions. Thus, we would like to investigate why these point-based algorithms work well, whether there are sub-classes of POMDPs that are computationally easier, and whether there are alternative measures that better capture the complexity of POMDP planning for point-based algorithms.

Our work is motivated by a benchmark problem called Tag [11], in which a robot needs to search and tag a moving target that tends to move away from it. The environment is modeled as a grid. The robot's position is fully observable. The target's position is not observable, *i.e.*, unknown to the robot, unless the target is in the same grid position as the robot. The joint state of the robot and target positions is thus only partially observable. The problem has 870 states in total, resulting in a belief space of 870 dimensions. Tag was introduced in the work on Point-Based Value Iteration (PBVI) [11], one of the first point-based POMDP algorithms. At the time, it was among the largest POMDP problems ever attempted and was considered a challenge for fast, scalable POMDP algorithms [11]. Surprisingly, only two years later, another point-based algorithm [14] computed an approximate solution to Tag, a problem with an 870-dimensional belief space, in less than a minute!

One important feature that underlies the success of many point-based algorithms is that they only explore a subset $\mathcal{R}(b_0) \subseteq \mathcal{B}$, usually called the *reachable space* from $b_0$. The reachable space $\mathcal{R}(b_0)$

contains all points reachable from a given initial belief point $b_0 \in \mathcal{B}$ under arbitrary sequences of actions and observations. One may then speculate that the reason for point-based algorithms' good performance on Tag is that its reachable space $\mathcal{R}(b_0)$ has much lower dimensionality than $\mathcal{B}$. This is, however, not true. By checking the dimensionality of a large set of points sampled from $\mathcal{R}(b_0)$, we have found that the dimensionality of $\mathcal{R}(b_0)$ is at least 860 and thus almost as large as $\mathcal{B}$.

In this paper, we propose to use the *covering number* as an alternative measure of the complexity of POMDP planning ( Section 4). Intuitively, the covering number of a space is the minimum number of given size balls that needed to cover the space fully. We show that an approximately optimal POMDP solution can be computed in time polynomial in the covering number of $\mathcal{R}(b_0)$. The covering number also reveals that the belief space for Tag behaves more like the union of some 29-dimensional spaces rather than an 870-dimensional space, as the robot's position is fully observed. Therefore, Tag is probably not as hard as it was thought to be, and the covering number captures the complexity of the Tag problem better than the dimensionality of the belief space (the number of states) or the dimensionality of the reachable space.

We further ask whether it is possible to compute an approximate solution efficiently under the weaker condition of having a small covering number for an optimal reachable $\mathcal{R}^*(b_0)$, which contains only points in $\mathcal{B}$ reachable from $b_0$ under an optimal policy. Unfortunately, we can show that this problem is NP-hard. The problem remains NP-hard, even if the optimal policies have a compact piecewise-linear representation using $\alpha$-vectors. However, we can also show that given a suitable set of points that "cover" $\mathcal{R}^*(b_0)$ well, a good approximate solution can be computed in polynomial time. Together, the negative and the positive results indicate that using sampling to approximate an optimal reachable space, and not just the reachable space, may be a promising approach in practice. We have already obtained initial experimental evidence that supports this idea. Through careful sampling and pruning, our new point-based algorithm solved the Tag problem in less than 5 seconds [4].

The covering number highlights several properties that reduce the complexity of POMDP planning in practice, and it helps to quantify their effects (Section 5). Highly informative observations usually result in beliefs with sparse support and substantially reduce the covering number. For example, fully observed state variables reduce the covering number by a doubly exponential factor. Interestingly, smooth beliefs, usually a result of imperfect actions and uninformative observations, also reduce the covering number. In addition, state-transition matrices with special structures, such as *circulant matrices* [1], restrict the space of reachable beliefs and reduce the covering number correspondingly.

## 2  Related Works

POMDPs provide a principled mathematical framework for planning and decision-making under uncertainty [13, 5], but they are notoriously hard to solve [10, 7, 9, 8]. It has been shown that finding an optimal policy over the entire belief space for a finite-horizon POMDP is PSPACE-complete [10] and that finding an optimal policy over an infinite horizon is undecidable [9].

As a result, there has been a lot of work on computing approximate POMDP solutions [2], including a number of point-based POMDP algorithms [16, 11, 15, 14, 3]. Some point-based algorithms were able to compute reasonably good policies for very large POMDPs with hundreds of thousands states. The success of these algorithms motivated us to try to understand why and when they work well.

The approximation errors of some point-based algorithms have been analyzed [11, 14], but these analyses do not address the general question of when an approximately optimal policy can be computed efficiently in polynomial time. We provide both positive and negative results showing the difficulty of computing approximate POMDP solutions. The proof techniques used for Theorems 1 and 2 are similar to those used for analyzing an approximation algorithm for large (fully observable) MDPs [6]. While the algorithm in [6] handles large state spaces well, it does not run in polynomial time: it appears that additional assumptions such as those made in this paper are required for polynomial time results. Our hardness result is closely related to that for finite-horizon POMDPs [8], but we give a direct reduction from the Hamiltonian cycle problem.

## 3  Preliminaries

A POMDP models an agent taking a sequence of actions under uncertainty to maximize its total reward. Formally it is specified as a tuple $(S, A, O, T, Z, R, \gamma)$, where $S$ is a set of discrete states, $A$ is a finite set of actions, and $O$ is a set of discrete observations. At each time step, the agent

takes some action $a \in A$ and moves from a start state $s$ to an end state $s'$. The end state $s'$ is given by a state-transition function $T(s, a, s') = p(s'|s, a)$, which gives the probability that the agent lies in $s'$, after taking action $a$ in state $s$. The agent then makes an observation to gather information on its current state. The outcome of observing $o \in O$ is given by an observation function $Z(s, a, o) = p(o|s, a)$ for $s \in S$ and $a \in A$. The reward function $R$ gives the agent a real-valued reward $R(s, a)$ if it takes action $a$ in state $s$, and the goal of the agent is to maximize its expected total reward by choosing a suitable sequence of actions. In this paper, we consider only infinite-horizon POMDPs with discounted reward. Thus, the expected total reward is given by $\mathrm{E}[\sum_{t=0}^{\infty} \gamma^t R(s_t, a_t)]$, where $\gamma \in (0, 1)$ is a discount factor, and $s_t$ and $a_t$ denote the agent's state and the action at time $t$.

Since the agent's state is only partially observable, we rely on the concept of a belief, which is simply a probability distribution over $S$, represented disretely as a vector.

A POMDP solution is a policy $\pi$ that specifies the action $\pi(b)$ for every belief $b$. Our goal is to find an optimal policy $\pi^*$ that maximizes the expected total reward. A policy $\pi$ induces a value function $V^\pi$ that specifies the value $V^\pi(b)$ of every belief $b$ under $\pi$. It is known that $V^*$, the value function associated the optimal policy $\pi^*$, can be approximated arbitrarily closely by a convex, piecewise-linear function $V(b) = \max_{\alpha \in \Gamma}(\alpha \cdot b)$, where $\Gamma$ is a finite set of vectors called $\alpha$-vectors.

The optimal value function $V^*$ satisfies the following Lipschitz condition:

**Lemma 1** *For any two belief points $b$ and $b'$, if $||b - b'|| \leq \delta$, then $|V^*(b) - V^*(b')| \leq \frac{R_{\max}}{1-\gamma}\delta$.*[1]

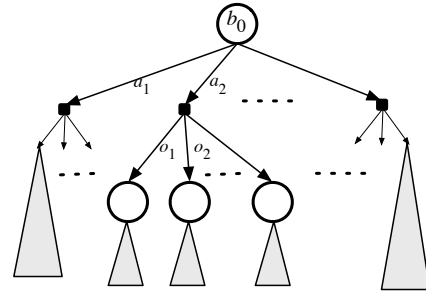

Throughout this paper, we always use the $l_1$ metric to measure the distance between belief points: for $b, b' \in \mathbb{R}^d$, $||b - b'|| = \sum_{i=1}^{d} |b_i - b'_i|$. The Lipschitz condition bounds the change of a value function using the distance between belief points. It provides the basis for approximating the value at a belief point by the values of other belief points nearby.

To find an approximately optimal policy, point-based algorithms explore only the reachable belief space $\mathcal{R}(b_0)$ from a given initial belief point $b_0$. Strictly speaking, these algorithms compute only a policy over $\mathcal{R}(b_0)$, rather than the entire belief space

Figure 1: The belief tree rooted at $b_0$.

$\mathcal{B}$. We can view the exploration of $\mathcal{R}(b_0)$ as searching a *belief tree* $T_\mathcal{R}$ rooted at $b_0$ (Figure 1). The nodes of $T_\mathcal{R}$ correspond to beliefs in $\mathcal{R}(b_0)$. The edges correspond to action-observation pairs. Suppose that a child node $b'$ is connected to its parent $b$ by an edge $(a, o)$. We can compute $b'$ using the formula $b'(s') = \tau(b, a, o) = \eta Z(s', a, o) \sum_s T(s, a, s')b(s)$, where $\eta$ is a normalizing constant. After obtaining enough belief points from $\mathcal{R}(b_0)$, point-based algorithms perform backup operations over them to compute an approximately optimal value function.

## 4 The Covering Number and the Complexity of POMDP Planning

Our first goal is to show that if the covering number of a reachable space $\mathcal{R}(b_0)$ is small, then an approximately optimal policy in $\mathcal{R}(b_0)$ can be computed efficiently. We start with the definition of the covering number:

**Definition 1** Given a metric space $X$, a $\delta$-*cover* of a set $B \subseteq X$ is a set of point $C \subseteq X$ such that for every point $b \in B$, there is a point $c \in C$ with $||b - c|| < \delta$. If all the points in $C$ also lie in $B$, then we say that $C$ is a *proper cover* of $B$. The $\delta$-*covering number* of $B$, denoted by $\mathcal{C}(\delta)$, is the size of the smallest $\delta$-cover of $B$.

Intuitively, the covering number is equal to the minimum number of balls of radius $\delta$ needed to cover the set $B$. A closely related notion is that of the packing number:

**Definition 2** Given a metric space $X$, a $\delta$-*packing* of a set $B \subseteq X$ is a set of points $P \subseteq B$ such that for any two points $p_1, p_2 \in P$, $||p_1 - p_2|| \geq \delta$. The $\delta$-packing number of a set $B$, denoted by $\mathcal{P}(\delta)$, is the size of the largest $\delta$-packing of $B$.

For any set $B$, the following relationship holds between packing and covering numbers.

**Lemma 2** $\mathcal{C}(\delta) \leq \mathcal{P}(\delta) \leq \mathcal{C}(\delta/2)$.

We are now ready to state our first main result. It shows that for any point $b_0 \in \mathcal{B}$, if the covering number of $\mathcal{R}(b_0)$ grows polynomially with the parameters of interest, then a good approximation of the value at $b_0$ can be computed in polynomial time.

**Theorem 1** *For any $b_0 \in \mathcal{B}$, let $\mathcal{C}(\delta)$ be the $\delta$-covering number of $\mathcal{R}(b_0)$. Given any constant $\epsilon > 0$, an approximation $V(b_0)$ of $V^*(b_0)$, with error $|V^*(b_0) - V(b_0)| \leq \epsilon$, can be found in time*

$$O\left( \mathcal{C}\left( \frac{(1-\gamma)^2 \epsilon}{4\gamma R_{\max}} \right)^2 \log_\gamma \frac{(1-\gamma)\epsilon}{2R_{\max}} \right).$$

**Proof.** To prove the result, we give an algorithm that computes the required approximation. It performs a depth-first search on a depth-bounded belief tree and uses approximate memorization to avoid unnecessarily computing the values of very similar beliefs. Intuitively, to achieve a polynomial time algorithm, we bound the height of the tree by exploiting the discount factor and bound the width of the tree by exploiting the covering number.

We perform the depth-first search recursively on a belief tree $T_\mathcal{R}$ that has root $b_0$ and height $h$, while maintaining a $\delta$-packing of $\mathcal{R}(b_0)$ at *every* level of $T_\mathcal{R}$. Suppose that the search encounters a new belief node $b$ at level $i$ of $T_\mathcal{R}$. If $b$ is within a distance $\delta$ of a point $b'$ in the current packing at level $i$, we set $V(b) = V(b')$, abort the recursion at $b$, and backtrack. Otherwise, we recursively search the children of $b$. When the search returns, we perform a backup operation to compute $V(b)$ and add $b$ to the packing at level $i$. If $b$ is a leaf node of $T_\mathcal{R}$, we set $V(b) = 0$. We build a separate packing at each level of $T_\mathcal{R}$, as each level has a different approximation error.

We now calculate the values for $h$ and $\delta$ required to achieve the given approximation bound $\epsilon$ at $b_0$. Let $\epsilon_i = |V^*(b) - V(b)|$ denote the approximation error for a node $b$ at level $i$ of $T_\mathcal{R}$, if the recursive search continues in the children of $b$. By convention, the leaf nodes are at level 0. Similarly, let $\epsilon'_i$ denote the error for $b$, if the search aborts at $b$ and $V(b) = V(b')$ for some $b'$ in the packing at level $i$. Hence,

$$
\begin{aligned}
\epsilon'_i &= |V^*(b) - V(b')| \\
&\leq |V^*(b) - V^*(b')| + |V^*(b') - V(b')| \\
&\leq \frac{R_{\max}}{1-\gamma}\delta + \epsilon_i,
\end{aligned}
$$

where the last inequality uses Lemma 1 and the definition of $\epsilon_i$. Clearly, $\epsilon_0 \leq R_{\max}/(1-\gamma)$. To calculate $\epsilon_i$ for a node $b$ at level $i$, we establish a recurrence. The children of $b$, which are at level $i-1$, have error at most $\epsilon'_{i-1}$. Since a backup operation is performed at $b$, we have $\epsilon_i \leq \gamma \epsilon'_{i-1}$ and thus the recurrence $\epsilon_i \leq \gamma(\epsilon_{i-1} + \frac{R_{\max}}{1-\gamma}\delta)$. Expanding the recurrence, we find that the error $\epsilon_h$ at the root $b_0$ is given by

$$
\begin{aligned}
|V^*(b_0) - V(b_0)| &\leq \frac{\gamma R_{\max}(1-\gamma^h)}{(1-\gamma)^2}\delta + \gamma^h \frac{R_{\max}}{1-\gamma} \\
&\leq \frac{\gamma R_{\max}}{(1-\gamma)^2}\delta + \gamma^h \frac{R_{\max}}{1-\gamma}.
\end{aligned}
$$

By setting $\delta = \frac{(1-\gamma)^2 \epsilon}{2\gamma R_{\max}}$ and $h = \log_\gamma \frac{(1-\gamma)\epsilon}{2R_{\max}}$, we can guarantee $|V^*(b_0) - V(b_0)| \leq \epsilon$.

We now work out the running time of the algorithm. For each node $b$ in the packings, the algorithm expands it by calculating the beliefs and the corresponding values for all its children and performing a backup operation at $b$ to compute $V(b)$. It takes $O(|S|^2)$ time to calculate the belief at a child node. We then perform a nearest neighbor search in $O(\mathcal{P}(\delta)|S|)$ time to check whether the child node lies within a distance $\delta$ of any point in the packing at that level. Since $b$ has $|A||O|$ children, the expansion operation takes $O(|A||O||S|(|S| + \mathcal{P}(\delta)))$ time. The backup operation then computes $V(b)$ as an average of its children's values, weighted by the probabilities specified by the observation function, and takes only $O(|A||O|)$ time. Since there are $h$ packings of size $\mathcal{P}(\delta)$ each and by Lemma 2, $\mathcal{P}(\delta) \leq \mathcal{C}(\delta/2)$, the total running time of our algorithm is given by

$$O\left( h\mathcal{C}(\delta/2)|A||O||S|(|S| + \mathcal{C}(\delta/2)) \right).$$

We assume that $|S|$, $|A|$, and $|O|$ are constant to focus on the dependency on the covering number, and the above expression then becomes $O(h\mathcal{C}(\delta/2)^2)$. Substituting in the values for $h$ and $\delta$, we get the final result. $\square$

The algorithm in the above proof can be used on-line to choose an approximately optimal action at $b_0$. We first estimate the values for all the child nodes of $b_0$ and then select the action resulting in the highest value. Suppose that at each belief point reachable from $b_0$, we perform such an on-line search for action selection. Using the technique in [12], one can show that if the value function approximations at all the child nodes have error at most $\epsilon$, then the policy $\pi$ implicitly defined by the on-line search has approximation error $|V^*(b) - V^\pi(b)| \le 2\gamma\epsilon/(1-\gamma)$ for all $b$ in $\mathcal{R}(b_0)$.

Instead of performing the on-line search, one may want to precompute an approximately optimal value function over $\mathcal{R}(b_0)$ and perform one-step look-ahead on it at runtime for action selection. The algorithm in Theorem 1 is not sufficient for this purpose, as it samples only enough points from $\mathcal{R}(b_0)$ to give a good value estimate at $b_0$, but the sampled points do not form a cover of $\mathcal{R}(b_0)$. One possibility would be to find a cover of $\mathcal{R}(b_0)$ first and then apply PBVI [11] over the points in the cover. Unfortunately, we do not know how to find a cover of $\mathcal{R}(b_0)$ efficiently. Instead, we give a randomized algorithm that computes an approximately optimal value function with high probability. Roughly, this algorithm incrementally builds a packing of $\mathcal{R}(b_0)$ at each level of $T_\mathcal{R}$. It first runs the algorithm in Theorem 1 to obtain an initial packing $P_i$ for each level $i$ and estimate the values of belief points in $P_i$. Then, to test whether the current packing $P_i$ covers $\mathcal{R}(b_0)$ well, it runs a set of simulations of a fixed size. If the simulations encounter new points not covered by $P_i$, we estimate their values and insert them into $P_i$. The process repeats until no more new belief points are discovered within a set of simulation. We show that if the set of simulations is sufficiently large, then the probability that in any future run of the policy, we encounter new belief points not covered by the final set of packings can be made arbitrarily small.

**Theorem 2** *For any $b_0 \in \mathcal{B}$, let $\mathcal{C}(\delta)$ be the $\delta$-covering number of $\mathcal{R}(b_0)$. Given constants $\beta \in (0, 1)$ and $\epsilon > 0$, a randomized algorithm can compute, with probability at least $1 - \beta$, an approximately optimal value function in time*

$$O\left(\frac{R_{\max}}{(1-\gamma)\epsilon}\left(\mathcal{C}\left(\frac{(1-\gamma)^3\epsilon}{16\gamma R_{\max}}\right)\log_\gamma\frac{(1-\gamma)\epsilon}{4R_{\max}}\right)^2\log\left(\frac{1}{\beta}\mathcal{C}\left(\frac{(1-\gamma)^3\epsilon}{16\gamma R_{\max}}\right)\log_\gamma\frac{(1-\gamma)\epsilon}{4R_{\max}}\right)\right).$$

*such that the one-step look-ahead policy $\pi$ induced by this value function has error $|V^*(b_0) - V^\pi(b_0)| \le \epsilon$. It takes $O\left(\mathcal{C}\left(\frac{(1-\gamma)^3\epsilon}{16\gamma R_{\max}}\right)\right)$ time to use this value function to select an action at runtime.*

Both theorems above assume tha a small covering number of $\mathcal{R}(b_0)$ for efficient computation. To relax this assumption, we may require only that the covering number for an optimal reachable space $\mathcal{R}^*(b_0)$ is small, as $\mathcal{R}^*(b_0)$ contains only points reachable under an optimal policy and can be much smaller than $\mathcal{R}(b_0)$. Unfortunately, under the relaxed condition, approximating the value at $b_0$ is NP-hard. We prove this by reduction from the Hamiltonian cycle problem. The main idea is to show that a Hamiltonian cycle exists in a given graph if and only an approximation to $V^*(b_0)$, with a suitably chosen error, can be computed for a POMDP whose optimal reachable space $\mathcal{R}^*(b_0)$ has a small covering number. The result is closely related to one for finite-horizon POMDPs [8].

**Theorem 3** *Given constant $\epsilon > 0$, computing an approximation $V(b_0)$ of $V^*(b_0)$, with error $|V(b_0) - V^*(b_0)| \le \epsilon|V^*(b_0)|$, is NP-hard, even if the covering number of $\mathcal{R}^*(b_0)$ is polynomial-sized.*

The result above assumes the standard encoding of POMDP input with state-transition functions, observation functions, and reward functions all represented discretely by matrices of suitable sizes.

By slightly extending the proof of Theorem 3, we can also show a related hardness result, which assumes that the optimal policy has a compact representation.

**Theorem 4** *Given constant $\epsilon > 0$, computing an approximation $V(b_0)$ of $V^*(b_0)$, with error $|V(b_0) - V^*(b_0)| \le \epsilon|V^*(b_0)|$, is NP-hard, even if the number of $\alpha$-vectors required to represent an optimal policy is polynomial-sized.*

On the other hand, if an oracle provides us a proper cover of an optimal reachable space $\mathcal{R}^*(b_0)$, then a good approximation of $V^*(b_0)$ can be found efficiently.

**Theorem 5** *For any $b_0 \in \mathcal{B}$, given constant $\epsilon > 0$ and a proper $\delta$-cover $C$ of $\mathcal{R}^*(b_0)$ with $\delta = \frac{(1-\gamma)^2 \epsilon}{2\gamma R_{\max}}$, an approximation $V(b_0)$ of $V^*(b_0)$, with error $|V^*(b_0) - V(b_0)| \leq \epsilon$, can be found in time*

$$O\left(|C|^2 + |C| \log_\gamma \frac{(1-\gamma)\epsilon}{2 R_{Max}}\right).$$

Together, the negative and the positive results (Theorems 3 to 5) indicate that a key difficulty for point-based algorithms lies in finding a cover for $\mathcal{R}^*(b_0)$. In practice, to overcome the difficulty, one may use problem-specific knowledge or heuristics to approximate $\mathcal{R}^*(b_0)$ through sampling.

Most point-based POMDP algorithms [11, 15, 14] interpolate the value function using $\alpha$-vectors. Although we use the nearest neighbor approximation to simplify the proofs of Theorems 1, 2, and 5, we want to point out that very similar results can be obtained using the $\alpha$-vector representation if we slightly modify the analysis of the approximation errors in the proofs.

# 5 Bounding the Covering Number

The covering number highlights several properties that reduce the complexity of POMDP planning in practice. We describe them below and show how they affect the covering number.

## 5.1 Fully Observed State Variables

Suppose that there are $d$ state variables, each of which has at most $k$ possible values. If $d'$ of these variables are fully observed, then for every such belief point, its vector representation contains at most $m = k^{d-d'}$ non-zero elements out of $k^d$ elements in total. For a given initial belief $b_0$, the belief vectors with the same non-zero pattern form a subspace in $\mathcal{R}(b_0)$, and $\mathcal{R}(b_0)$ is a union of these subspaces. We can compute a $\delta$-cover for each subspace by discretizing each non-zero element of the belief vectors to an accuracy of $\delta/m$, and the size of the resulting $\delta$-cover is at most $\left(\frac{m}{\delta}\right)^m$. There are $k^{d'}$ such subspaces. So the $\delta$-covering number of $\mathcal{R}(b_0)$ is at most $k^{d'} \left(\frac{m}{\delta}\right)^m = k^{d'} \left(\frac{k^{d-d'}}{\delta}\right)^{k^{d-d'}}$. The fully observed variables thus give a doubly exponential reduction in the covering number: it reduces the exponent by a factor of $k^{d'}$ at the cost of a multiplicative factor of $k^{d'}$.

**Proposition 1** *Suppose that a POMDP has $d$ state variables, each of which has at most $k$ possible values. If $d'$ state variables are fully observed, then for any belief point $b_0$, the $\delta$-covering number of the reachable belief space $\mathcal{R}(b_0)$ is at most $k^{d'} \left(\frac{k^{d-d'}}{\delta}\right)^{k^{d-d'}}$.*

Consider again the Tag problem described in Section 1. The state consists of both the robot's and the target's positions, as well as the status indicating whether the target is tagged. The robot and the target can occupy any position in an environment modeled as a grid of 29 cells. If the robot has the target tagged, they must be in the same position. So, there are $29 \times 29 + 29 = 870$ states in total, and the belief space $\mathcal{B}$ is 870-dimensional. However, the robot's position is fully observed. By Proposition 1, the $\delta$-covering number is at most $30 \cdot (30/\delta)^{30}$. Indeed, for Tag, any reachable belief space $\mathcal{R}(b_0)$ is effectively a union of two sets. One set corresponds to the case when the target is not tagged and consists of the union of 29 sub-spaces of 29 dimensions. The other set corresponds to the case when the target is tagged and consists of exactly 29 points. Clearly, the covering number captures the underlying complexity of $\mathcal{R}(b_0)$ more accurately than the dimensionality of $\mathcal{R}(b_0)$.

## 5.2 Sparse Beliefs

Highly informative observations often result in sparse beliefs, *i.e.*, beliefs whose vector representation is sparse. For example, in the Tag problem, the state is known exactly if the robot and the target are in the same position, leaving only a single non-zero element in the belief vector. Fully observed state variables usually result in very sparse beliefs and can be considered a special case.

If the beliefs are always sparse, we can exploit the sparsity to bound the covering number. Otherwise, sparsity may still give a hint that the covering number is smaller than what would be suggested by the dimensionality of the belief space. By exploiting the non-zeros patterns of belief vectors in a way similar to that in Section 5.1, we can derive the following result:

**Proposition 2** *Let $B$ be a set in an $n$-dimensional belief space. If every belief in $B$ can be represented as a vector with at most $m$ non-zero elements, then the $\delta$-covering number of $B$ is $O(n^m \left(\frac{m}{\delta}\right)^m)$.*

### 5.3 Smooth Beliefs

Sparse beliefs are often peaky. Interestingly, when the beliefs are sufficiently smooth, *e.g.*, when their Fourier representations are sparse, the covering number is also small. Below we give a more general result, assuming that the beliefs can be represented as a linear combination of a small number of basis vectors.

**Proposition 3** *Let $B$ be a set in an $n$-dimensional belief space. Assume that every belief $b \in B$ can be represented as a linear combination of $m$ basis vectors such that the magnitudes of both the elements of the basis vectors and the coefficients representing $b$ are bounded by a constant $C$. The $\delta$-covering number of $B$ is $O((\frac{2C^2mn}{\delta})^m)$ when the basis vectors are real-valued, and $O((\frac{4C^2mn}{\delta})^{2m})$ when they are complex-valued.*

Smooth beliefs are usually a result of actions with high uncertainty and uninformative observations.

### 5.4 Circulant State-Transition Matrices

Let us now shift our attention from observations to actions, in particular, actions that can be represented by state-transition matrices with special structures. We start with an example. A mobile robot scout needs to navigate from a known start position to a goal position in a large environment modeled as a grid. It must not enter certain danger zones to avoid detection by enemies. The robot can take four actions to move in the $\{N, S, E, W\}$ directions, but have imperfect control. Since the environment is large, we assume that the robot always operates far away from the boundary and the boundary effect can be ignored. At each grid cell, the robot moves to the intended cell with probability $1 - p$ and moves diagonally to the two cells adjacent to the intended one with probability $0.5p$. The robot can use its sensors to make highly accurate observations on its current position, but by doing so, it runs the risk of being detected.

Under our assumptions, the state-transition functions representing robot actions are invariant over the grid cells and can thus be represented by *circulant matrices* [1]. Circulant matrices are widely used in signal processing and control theory, as they can represent all discrete-time linear translation-invariant systems. In the context of POMDPs, if applying a state-transition matrix to a belief $b$ corresponds to convolution with a suitable distribution, then the state-transition matrix is circulant. One of the key properties of circulant matrices is that they all share the same eigenvectors. Therefore, we can multiply them in any arbitrary order and obtain the same result. In our example, this means that given a set of robot moves, we can apply them in any order and the resulting belief on the robot's position is the same. This greatly reduces the number of possible beliefs and correspondingly the covering number in open-loop POMDPs, where there are no observations involved.

**Proposition 4** *Suppose that all $\ell$ state-transition matrices representing actions are circulant and that each matrix has at most $m$ eigenvalues whose magnitudes are greater than $\zeta$, with $0 < \zeta < 1$. In an open-loop POMDP, for any point $b_0$ in an $n$-dimensional belief space, the $\delta$-covering number of the reachable belief space $\mathcal{R}(b_0)$ is $O\left( \left(\frac{8\ell mn}{\delta}\right)^{2\ell m} + h^\ell \right)$, where $h = \log_\zeta(\delta/2n)$.*

In our example, suppose that the robot scout makes a sequences of moves and needs to decide when to take occasional observations along the way to localize itself. To bound the covering number, we divide the sequence of moves into subsequences such that each subsequence starts with an observation and ends right before the next observation. In each subsequence, the robot starts at a specific belief and moves without additional observations. So, within a subsequence, the beliefs encountered have a $\delta$-cover of size $O((8\ell mn/\delta)^{2\ell m} + h^\ell)$ by Proposition 4. Furthermore, since all the observations are highly informative, we assume that the initial beliefs of all subsequences can be represented as vectors with at most $m'$ non-zero elements. The set of all initial beliefs then has a $\delta$-cover of size $O(n^{m'}(m'/\delta)^{m'})$ by Proposition 2. From Lemma 3 below, we know that in an open-loop POMDP, two belief trajectories can only get closer to each other, as they progress.

**Lemma 3** *Let $M$ be a Markov matrix and $||b_1 - b_2|| \leq \delta$. Then $||Mb_1 - Mb_2|| \leq \delta$.*

Therefore, to get a $\delta$-cover of the space $\mathcal{R}(b_0)$ that the robot scout can reach from a given $b_0$, it suffices to first compute a $\delta/2$-cover $C$ of the initial belief points for all possible subsequences of moves and then take the union of the $\delta/2$-covers of the belief points traversed by the subsequences whose initial belief points lie in $C$. The $\delta$-cover of $\mathcal{R}(b_0)$ then has its size bounded by $O(n^{m'}(2m'/\delta)^{m'}(16\ell mn/\delta)^{2\ell m} + h^\ell)$, where $h = \log_\zeta(\delta/4n)$.

The requirement of translation invariance means that circulant matrices have some limitations in modeling certain phenomena well. In mobile robot navigation, obstacles or boundaries in the environment often cause difficulties. However, if the environment is sufficiently large and the obstacles are sparse, the behaviors of some systems can be approximated by circulant matrices.

## 6 Conclusion

We propose the covering number as a measure of the complexity of POMDP planning. We believe that for point-based algorithms, the covering number captures the difficulty of computing approximate solutions to POMDPs better than other commonly used measures, such as the number of states. The covering number highlights several interesting properties that reduce the complexity of POMDP planning, and quantifies their effects. Using the covering number, we have shown several results that help to identify the main difficulty of POMDP planning using point-based algorithms. These results indicate that a promising approach in practice is to approximate an optimal reachable space through sampling. We are currently exploring this idea and have already obtained promising initial results [4]. On a set of standard test problems, our new point-based algorithm outperformed the fastest existing point-based algorithm by 5 to 10 times on some problems, while remaining competitive on others.

**Acknowledgements.** We thank Leslie Kaelbling and Tomás Lozano-Pérez for many insightful discussions on POMDPs. This work is supported in part by NUS ARF grants R-252-000-240-112 and R-252-000-243-112.

## Footnotes

[1]The proofs of this and other results are available as an appendix at `http://motion.comp.nus.edu.sg/papers/nips07.pdf`.

## References

[1] R.M. Gray. *Toeplitz and Circulant Matrices: A Review*. Now Publishers Inc, 2006.

[2] M. Hauskrecht. Value-function approximations for partially observable Markov decision processes. *J. Artificial Intelligence Research*, 13:33–94, 2000.

[3] J. Hoey, A. von Bertoldi, P. Poupart, and A. Mihailidis. Assisting persons with dementia during handwashing using a partially observable Markov decision process. In *Proc. Int. Conf. on Vision Systems*, 2007.

[4] D. Hsu, W.S. Lee, and N. Rong. Accelerating point-based POMDP algorithms through successive approximations of the optimal reachable space. Technical Report TRA4/07, National University of Singapore. School of Computing, April 2007.

[5] L.P. Kaelbling, M.L. Littman, and A.R. Cassandra. Planning and acting in partially observable stochastic domains. *Artificial Intelligence*, 101(1–2):99–134, 1998.

[6] M. Kearns, Y. Mansour, and A.Y. Ng. A sparse sampling algorithm for near optimal planning in large Markov decision processes. *Machine Learning*, 49(2-3):193–208, 2002.

[7] M.L. Littman. *Algorithms for sequential decision making*. PhD thesis, Dept. of Computer Science, Brown University, 1996.

[8] C. Lusena, J. Goldsmith, and M. Mundhenk. Nonapproximability results for partially observable Markov decision processes. *J. Artificial Intelligence Research*, 14:83–103, 2002.

[9] O. Madani, S. Hanks, and A. Condon. On the undecidability of probabilistic planning and infinite-horizon partially observable Markov decision problems. In *Proc. Nat. Conf. on Artificial Intelligence*, pages 541–548, 1999.

[10] C. Papadimitriou and J.N. Tsisiklis. The complexity of Markov decision processes. *Mathematics of Operations Research*, 12(3):441–450, 1987.

[11] J. Pineau, G. Gordon, and S. Thrun. Point-based value iteration: An anytime algorithm for POMDPs. In *Proc. Int. Jnt. Conf. on Artificial Intelligence*, pages 477–484, 2003.

[12] S.P. Singh and R.C. Yee. An upper bound on the loss from approximate optimal-value functions. *Machine Learning*, 16(3):227–233, 1994.

[13] R.D. Smallwood and E.J. Sondik. The optimal control of partially observable Markov processes over a finite horizon. *Operations Research*, 21:1071–1088, 1973.

[14] T. Smith and R. Simmons. Point-based POMDP algorithms: Improved analysis and implementation. In *Proc. Uncertainty in Artificial Intelligence*, 2005.

[15] M.T.J. Spaan and N. Vlassis. A point-based POMDP algorithm for robot planning. In *Proc. IEEE Int. Conf. on Robotics & Automation*, 2004.

[16] N.L. Zhang and W. Zhang. Speeding up the convergence of value iteration in partially observable Markov decision processes. *Journal of Artificial Intelligence Research*, 14:29–51, 2002.